# A Rapid Graph-based Method for Arbitrary Transformation-Invariant Pattern Classification

**Alessandro Sperduti**
Dipartimento di Informatica
Università di Pisa
Corso Italia 40
56125 Pisa, ITALY
perso@di.unipi.it

**David G. Stork**
Machine Learning and Perception Group
Ricoh California Research Center
2882 Sand Hill Road #115
Menlo Park, CA USA 94025-7022
stork@crc.ricoh.com

## Abstract

We present a graph-based method for rapid, accurate search through prototypes for transformation-invariant pattern classification. Our method has in theory the same recognition accuracy as other recent methods based on "tangent distance" [Simard et al., 1994], since it uses the same categorization rule. Nevertheless ours is significantly faster during classification because far fewer tangent distances need be computed. Crucial to the success of our system are 1) a novel graph architecture in which transformation constraints and geometric relationships among prototypes are encoded during learning, and 2) an improved graph search criterion, used during classification. These architectural insights are applicable to a wide range of problem domains. Here we demonstrate that on a handwriting recognition task, a basic implementation of our system requires less than half the computation of the Euclidean sorting method.

## 1   INTRODUCTION

In recent years, the crucial issue of incorporating invariances into networks for pattern recognition has received increased attention, most especially due to the work of

Simard and his colleagues. To a regular hierachical backpropagation network Simard et al. [1992] added a Jacobian network, which insured that directional derivatives were also learned. Such derivatives represented directions in feature space corresponding to the invariances of interest, such as rotation, translation, scaling and even line thinning. On small training sets for a function approximation problem, this hybrid network showed performance superior to that of a highly tuned backpropagation network taken alone; however there was negligible improvement on large sets. In order to find a simpler method applicable to real-world problems, Simard, Le Cun & Denker [1993] later used a variation of the nearest neighbor algorithm, one incorporating "tangent distance" ($T$-distance or $D_T$) as the classification metric — the smallest Euclidean distance between patterns after the optimal transformation. In this way, state-of-the-art accuracy was achieved on an isolated handwritten character task, though at quite high computational complexity, owing to the inefficient search and large number of Euclidean and tangent distances that had to be calculated.

Whereas Simard, Hastie & Saeckinger [1994] have recently sought to reduce this complexity by means of pre-clustering stored prototypes, we here take a different approach, one in which a (graph) data structure formed during learning contains information about transformations and geometrical relations among prototypes. Nevertheless, it should be noted that our method can be applied to a reduced (clustered) training set such as they formed, yielding yet faster recognition. Simard [1994] recently introduced a hierarchical structure of successively lower resolution patterns, which speeds search only if a minority of patterns are classified more accurately by using the tangent metric than by other metrics. In contrast, our method shows significant improvement even if the majority or all of the patterns are most accurately classified using the tangent distance.

Other methods seeking fast invariant classification include Wilensky and Manukian's scheme [1994]. While quite rapid during recall, it is more properly considered *distortion* (rather than coherent transformation) invariant. Moreover, some transformations such as line thinning cannot be naturally incorporated into their scheme. Finally, it appears as if their scheme scales poorly (compared to tangent metric methods) as the number of invariances is increased.

It seems somewhat futile to try to improve significantly upon the recognition *accuracy* of the tangent metric approach — for databases such as NIST isolated handwritten characters, Simard et al. [1993] reported accuracies matching that of *humans*! Nevertheless, there remains much that can be done to increase the computational efficiency during recall. This is the problem we address.

## 2   TRANSFORMATION INVARIANCE

In broad overview, during learning our method constructs a labelled graph data structure in which each node represents a stored prototype (labelled by its category) as given by a training set, linked by arcs representing the $T$-distance between them. Search through this graph (for classification) takes advantage of the graph structure and an improved search criterion. To understand the underlying computations, we must first consider tangent space.

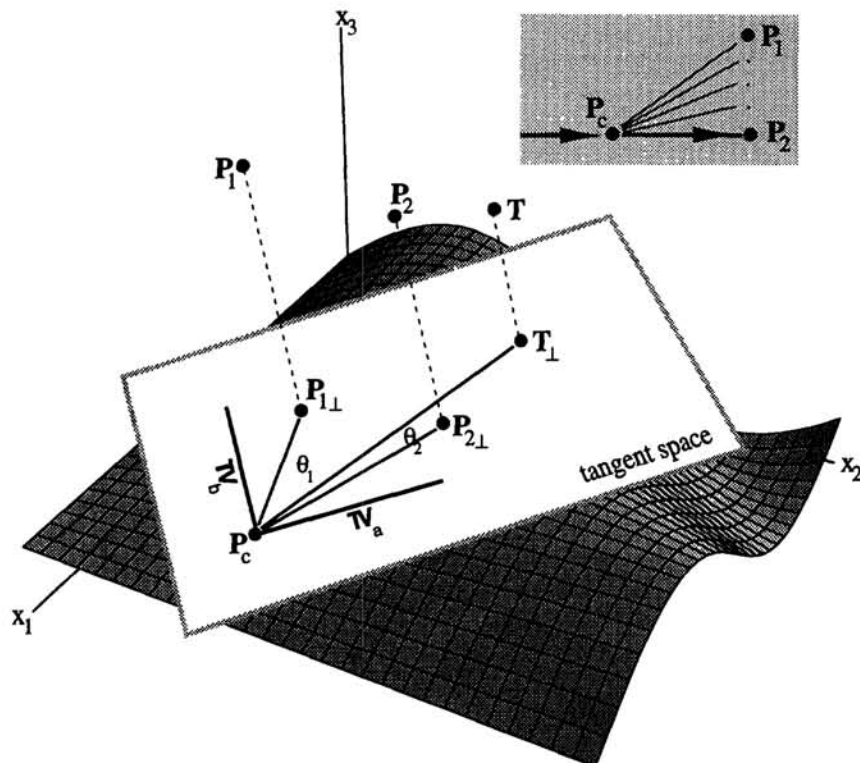

Figure 1: Geometry of tangent space. Here, a three-dimensional feature space contains the "current" prototype, $\mathbf{P}_c$, and the subspace consisting of all patterns obtainable by performing continuous transformations of it (shaded). Two candidate prototypes and a test pattern, $\mathbf{T}$, as well as their projections onto the $T$-space of $\mathbf{P}_c$ are shown. The insert (above) shows the progression of search through the corresponding portion of the recognition graph. The goal is to rapidly find the prototype closest to $\mathbf{T}$ (in the $T$-distance sense), and our algorithm (guided by the minimum angle $\theta_j$ in the tangent space) finds that $\mathbf{P}_2$ is so closer to $\mathbf{T}$ than are either $\mathbf{P}_1$ or $\mathbf{P}_c$ (see text).

Figure 1 illustrates geometry of tangent space and the relationships among the fundamental entities in our trained system. A labelled ("current") trained pattern is represented by $\mathbf{P}_c$, and the (shaded) surface corresponds to patterns arising under continuous transformations of $\mathbf{P}_c$. Such transformations might include rotation, translation, scaling, line thinning, etc. Following Simard et al. [1993], we approximate this surface in the vicinity of $\mathbf{P}_c$ by a subspace — the tangent space or $T$-space of $\mathbf{P}_c$ — which is spanned by "tangent" vectors, whose directions are determined by infinitessimally transforming the prototype $\mathbf{P}_c$. The figure shows an ortho-normal basis $\{\mathsf{TV}_a, \mathsf{TV}_b\}$, which helps to speed search during classification, as we shall see. A test pattern $\mathbf{T}$ and two other (candidate) prototypes as well as their projections onto the $T$-space of $\mathbf{P}_c$ are shown.

## 3   THE ALGORITHMS

Our overall approach includes constructing a graph (during learning), and searching it (for classification). The graph is constructed by the following algorithm:

---

**Graph construction**

**Initialize** $N =$ # patterns; $k =$ # nearest neighbors; $t =$ # invariant transformations

**Begin Loop** For each prototype $\mathbf{P}_i$ ($i = 1 \rightarrow N$)

- Compute a $t$-dimensional orthonormal basis for the $T$-space of $\mathbf{P}_i$
- Compute ("one-sided") $T$-distance of each of the $N - 1$ prototypes $\mathbf{P}_j$ ($j \neq i$) using $\mathbf{P}_i$'s $T$-space
- Represent $\mathbf{P}_{j\perp}$ (the projection of $\mathbf{P}_j$ onto the $T$-space of $\mathbf{P}_i$) in the tangent orthonormal frame of $\mathbf{P}_i$
- Connect $\mathbf{P}_i$ to each of its $k$ $T$-nearest neighbors, storing their associated normalized projections $\mathbf{P}_{j\perp}^*$

**End Loop**

---

During classification, our algorithm permits rapid search through prototypes. Thus in Figure 1, starting at $\mathbf{P}_c$ we seek to find another prototype (here, $\mathbf{P}_2$) that is closer to the test point $\mathbf{T}$. After $\mathbf{P}_2$ is so chosen, *it* becomes the current pattern, and the search is extended using *its* $T$-space. Graph search ends when the closest prototype to $\mathbf{T}$ is found (i.e., closest in a $T$-distance sense).

We let $D_T^c$ denote the current minimum tangent distance. Our search algorithm is:

---

**Graph search**

**Input** Test pattern $\mathbf{T}$

**Initialize**

- Choose initial candidate prototype, $\mathbf{P}_o$
- Set $\mathbf{P}_c \leftarrow \mathbf{P}_o$
- Set $D_T^c \leftarrow D_T(\mathbf{P}_c, \mathbf{T})$, i.e., the $T$-distance of $\mathbf{T}$ from $\mathbf{P}_c$

**Do**

- For each prototype $\mathbf{P}_j$ connected to $\mathbf{P}_c$ compute $cos(\theta_j) = \frac{T_\perp \cdot P_{j\perp}^*}{|T_\perp|}$
- Sort these prototypes by increasing values of $\theta_j$ and put them into a candidate list
- Pick $\mathbf{P}_j$ from the top of the candidate list
- In $T$-space of $\mathbf{P}_j$, compute $D_T(\mathbf{P}_j, \mathbf{T})$

  **If** $D_T(\mathbf{P}_j, \mathbf{T}) < D_T^c$ **then** $\mathbf{P}_c \leftarrow \mathbf{P}_j$ and $D_T^c \leftarrow D_T(\mathbf{P}_j, \mathbf{T})$
  **otherwise** mark $\mathbf{P}_j$ as a "failure" (F), and pick next prototype from the candidate list

**Until** Candidate list empty

**Return** $D_T^c$ or the category label of the optimum prototype found

---

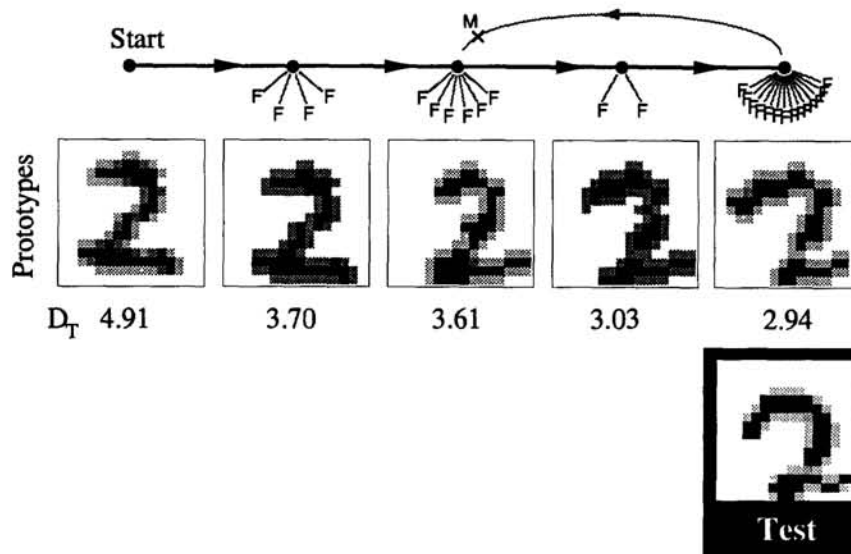

Figure 2: The search through the "2" category graph for the $T$-nearest stored prototype to the test pattern is shown ($N = 720$ and $k = 15$ nearest neighbors). The number of $T$-distance calculations is equal to the number of nodes visited plus the number of failures (marked F); i.e., in the case shown $5 + 26 = 31$. The backward search step attempt is thwarted because the middle node has already been visited (marked M). Notice in the prototypes how the search is first a downward shift, then a counter-clockwise rotation — a mere four steps through the graph.

Figure 2 illustrates search through a network of "2" prototypes. Note how the $T$-distance of the test pattern decreases, and that with only four steps through the graph the optimal prototype is found.

There are several ways in which our search technique can be incorporated into a classifier. One is to store all prototypes, regardless of class, in a single large graph and perform the search; the test pattern is classified by the label of the optimal prototype found. Another, is to employ *separate* graphs, one for each category, and search through them (possibly in parallel); the test is classified by the minimum $T$-distance prototype found. The choice of method depends upon the hardware limitations, performance speed requirements, etc. Figure 3 illustrates such a search through a "2" category graph for the closest prototype to a test pattern "5." We report below results using a single graph per category, however.

## 3.1 Computational complexity

If a graph contains $N$ prototypes with $k$ pointers (arcs) each, and if the patterns are of dimension $m$, then the storage requirement is $O(N((t+1) \cdot m^2 + kt))$. The time complexity of training depends upon details of ortho-normalization, sorting, etc., and is of little interest anyway. Construction is more than an order of magnitude faster than neural network training on similar problems; for instance construction of a graph for $N = 720$ prototypes and $k = 100$ nearest neighbors takes less than

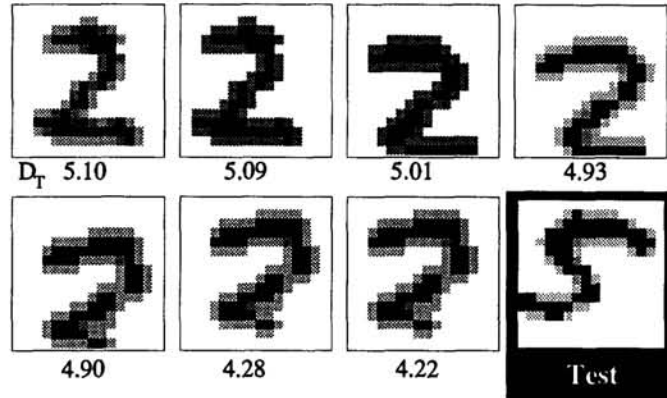

Figure 3: The search through a "2" category graph given a "5" test pattern. Note how the search first tries to find a prototype that matches the upper arc of the "5," and then one possessing skew or rotation. For this test pattern, the minimum $T$-distance found for the "5" category (3.62) is smaller than the one found for the "2" category shown here (4.22), and indeed for any other category. Thus the test pattern is correctly classified as a "5."

20 minutes on a Sparc 10.

The crucial quantity of interest is the time complexity for search. This is, of course, problem related, and depends upon the number of categories, transformation and prototypes and their statistical properties (see next Section). Worst case analyses (e.g., it is theoretically conceivable that nearly all prototypes must be visited) are irrelevant to practice.

We used a slightly non-obvious search criterion at each step, the function $cos(\theta_j)$, as shown in Figure 1. Not only could this criterion be calculated very efficiently in our orthonormal basis (by using simple inner products), but it actually led to a slightly more accurate search than Euclidean distance in the $T$-space — perhaps the most natural choice of criterion. The angle $\theta_j$ seems to guide the "flow" of the search along transformation directions toward the test point.

## 4   Simulations and results

We explored the search capabilities of our system on the binary handwritten digit database of Guyon, et al. [1991]. We needed to scale all patterns by a linear factor (0.833) to insure that rotated versions did not go outside the $16 \times 16$ pixel grid. As required in all $T$-space methods, the patterns must be continuous valued (i.e., here grayscale); this was achieved by convolution with a spatially symmetric Gaussian having $\sigma = .55$ pixels. We had 720 training examples in each of ten digit categories; the test set consisted of 1320 test patterns formed by transforming independent prototypes in all meaningful combinations of the $t = 6$ transformations (four spatial directions and two rotation senses).

We compared the Euclidean sorting method of Simard et al. [1993] to our graph

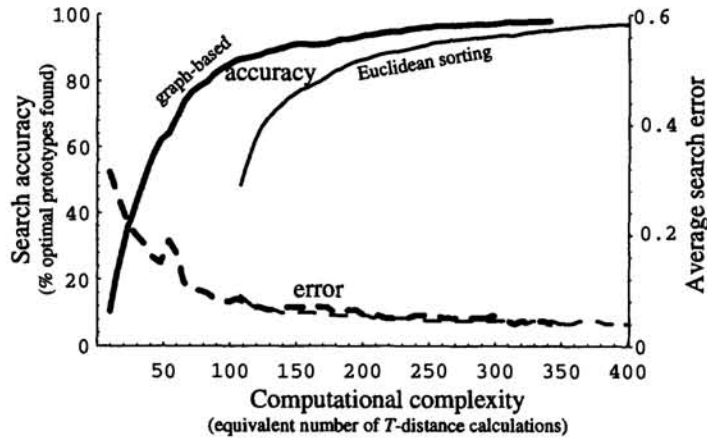

Figure 4: Comparison of graph-based (heavy lines) and standard Euclidean sorting searches (thin lines). Search accuracy is the percentage of optimal prototypes found on the full test set of 1320 patterns in a single category (solid lines). The average search error is the per pattern difference between the global optimum $T$-distance and the one actually found, averaged over the non-optimal prototypes found through the search (dashed lines). Note especially that for the same computational complexity, our method has the same average error, but that this average is taken over a much smaller number of (non-optimal) prototypes. For a given criterion search accuracy, our method requires significantly less computation. For instance, if 90% of the prototypes must be found for a requisite categorization accuracy (a typical value for asymptotically high recognition accuracy), our graph-based method requires less than half the computation of the Euclidean sorting method.

based method using the same data and transformations, over the full range of relevant computational complexities. Figure 4 summarizes our results. For our method, the computational complexity is adjusted by the number of neighbors inspected, $k$. For their Euclidean sorting method, it is adjusted by the percentage of Euclidean nearest neighbors that were then inspected for $T$-distance. We were quite careful to employ as many computational tricks and shortcuts on *both* methods we could think of. Our results reflect fairly on the full computational complexity, which was dominated by tangent and Euclidean distance calculations.

We note parenthetically that many of the recognition errors for both methods could be explained by the fact that we did not include the transformation of line thinning (solely because we lacked the preprocessing capabilities); the overall accuracy of both methods will increase when this invariance is also included.

## 5   CONCLUSIONS AND FUTURE WORK

We have demonstrated a graph-based method using tangent distance that permits search through prototypes significantly faster than the most popular current approach. Although not shown above, ours is also superior to other tree-based

methods, such as k-d-trees, which are less accurate. Since our primary concern was reducing the computational complexity of search (while matching Simard et al.'s accuracy), we have not optimized over preprocessing steps, such as the Gaussian kernel width or transformation set. We note again that our method can be applied to reduced training sets, for instance ones pruned by the method of Simard, Hastie & Saeckinger [1994]. Simard's [1994] recent method — in which *low-resolution* versions of training patterns are organized into a hierarchical data structure so as to reduce the number of multiply-accumulates required during search — is in some sense "orthogonal" to ours. Our graph-based method will work with his low-resolution images too, and thus these two methods can be unified into a hybrid system.

Perhaps most importantly, our work suggests a number of research avenues. We used just a single ("central") prototype $P_o$ to start search; presumably having several candidate starting points would be faster. Our general method may admit gradient descent learning of parameters of the search criterion. For instance, we can imagine scaling the different tangent basis vectors according to their relevance in guiding correct searches as determined using a validation set. Finally, our approach may admit elegant parallel implementations for real-world applications.

## Acknowledgements

This work was begun during a visit by Dr. Sperduti to Ricoh CRC. We thank I. Guyon for the use of her database of handwritten digits and Dr. K. V. Prasad for assistance in image processing.

## References

I. Guyon, P. Albrecht, Y. Le Cun, J. Denker & W. Hubbard. (1991) "Comparing different neural network architectures for classifying handwritten digits," *Proc. of the Inter. Joint Conference on Neural Networks*, vol. II, pp. 127-132, IEEE Press.

P. Simard. (1994) "Efficient computation of complex distance metrics using hierarchical filtering," in J. D. Cowan, G. Tesauro and J. Alspector (eds.) *Advances in Neural Information Processing Systems-6* Morgan Kaufmann pp. 168-175.

P. Simard, B. Victorrio, Y. Le Cun & J. Denker. (1992) "Tangent Prop — A formalism for specifying selected invariances in an adaptive network," in J. E. Moody, S. J. Hanson and R. P. Lippmann (eds.) *Advances in Neural Information Processing Systems-4* Morgan Kaufmann pp. 895-903.

P. Y. Simard, Y. Le Cun & J. Denker. (1993) "Efficient Pattern Recognition Using a New Transformation Distance," in S. J. Hanson, J. D. Cowan and C. L. Giles (eds.) *Advances in Neural Information Processing Systems-5* Morgan Kaufmann pp. 50-58.

P. Y. Simard, T. Hastie & E. Saeckinger. (1994) "Learning Prototype Models for Tangent Distance," *Neural Networks for Computing* Snowbird, UT (April, 1994).

G. D. Wilensky & N. Manukian. (1994) "Nearest Neighbor Networks: New Neural Architectures for Distortion-Insensitive Image Recognition," *Neural Networks for Computing* Snowbird, UT (April, 1994).